# Source Separation and Density Estimation by Faithful Equivariant SOM

**Juan K. Lin**
Department of Physics
University of Chicago
Chicago, IL 60637
jk-lin@uchicago.edu

**David G. Grier**
Department of Physics
University of Chicago
Chicago, IL 60637
d-grier@uchicago.edu

**Jack D. Cowan**
Department of Math
University of Chicago
Chicago, IL 60637
j-cowan@uchicago.edu

## Abstract

We couple the tasks of source separation and density estimation by extracting the local geometrical structure of distributions obtained from mixtures of statistically independent sources. Our modifications of the self–organizing map (SOM) algorithm results in purely digital learning rules which perform non–parametric histogram density estimation. The non–parametric nature of the separation allows for source separation of non–linear mixtures. An anisotropic coupling is introduced into our SOM with the role of aligning the network locally with the independent component contours. This approach provides an exact verification condition for source separation with no prior on the source distributions.

## 1 INTRODUCTION

Much of the current work on visual cortex modeling has focused on the generation of coding which captures statistical independence and sparseness (Bell and Sejnowski 1996, Olshausen and Field 1996). The Bell and Sejnowski model suffers from the parametric and intrinsically non–local nature of their source separation algorithm, while the Olshausen and Field model does not achieve true sparse–distributed coding where each cell has the same response probability (Field 1994). In this paper, we construct an extensively modified SOM with equipartition of activity as a steady–state for the task of local statistical independence processing and sparse–distributed coding.

Ritter and Schulten (1986) demonstrated that the density of the Kohonen SOM units is not proportional to the input density in the steady–state. In one dimension the Kohonen net under–represents high density and over–represents low density regions. Thus SOM's are generally not used for density estimation. Several modifications for controlling the magnification of the representation have appeared. Recently, Bauer et. al. (1996) used an "adaptive step size" , and Lin and Cowan (1996) used an $L_p$–norm weighting to control the magnification. Here we concentrate on the later's "faithful representation" algorithms for source separation and density estimation.

## 2 SHARPLY PEAKED DISTRIBUTIONS

Mixtures of sharply peaked source distributions will contain high density contours which correspond to the independent component contours. Blind separation can be performed rapidly for this case in a net with one dimensional branched topology. A digital learning rule where the updates only take on discrete values was used: [1]

$$\Delta \vec{w}_i = \kappa \Lambda(\epsilon) \cdot sgn(\vec{\xi} - \vec{w}_i), \tag{1}$$

where $\kappa$ is the learning rate, $\Lambda(\epsilon)$ the neighborhood function, $\{\vec{w}\}$ the SOM unit positions, and $\vec{\xi}$ the input.

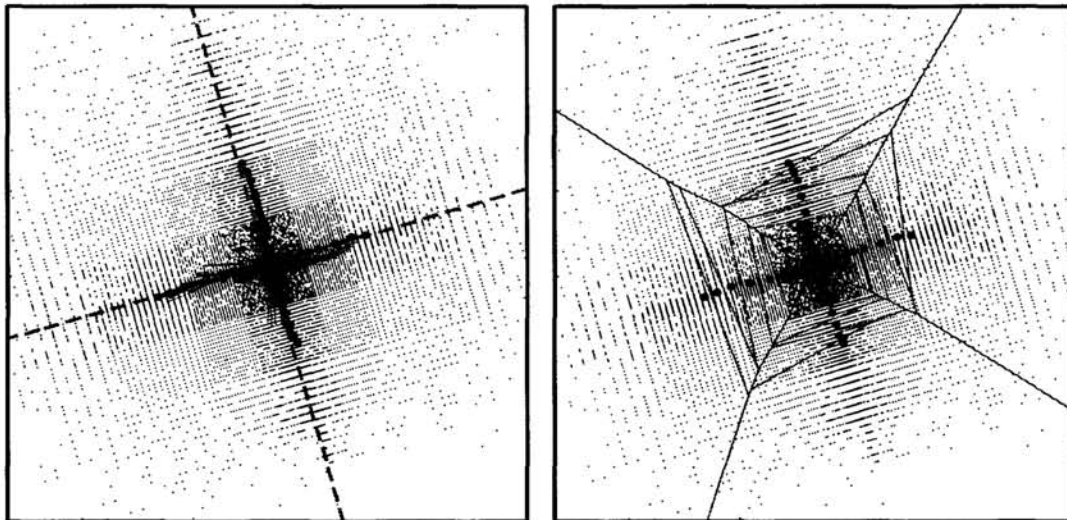

Figure 1: Left: linear source separation by branched net. Dashed lines correspond to the independent component axes. Net configuration is shown every 200 points. Dots denote the unit positions after 4000 points. Right: Voronoi partition of the vector space by the SOM units.

We performed source separation and coding of two mixed signals in a net with the topology of two cross-linked branches (see Fig. (1)). The neighborhood function

$\Lambda(\epsilon)$ is taken to be Gaussian where $\epsilon$ is the distance to the winning unit along the branch structure. Two speech audio files were randomly mixed and pre–whitened first to decorrelate the two mixtures. Since pre–whitening tends to orthogonalize the independent component axes, much of the processing that remains is rotation to find the independent component coordinate system. A typical simulation is shown in Fig. (1). The branches of the net quickly zero in on the high density directions. As seen from the nearest–neighbor Voronoi partition of the distribution (Fig. 1b), the branched SOM essentially performs a one dimensional equipartition of the mixture. The learning rule Eqn. 1 attempts to place each unit at the component-wise median of the distribution encompassed by its Voronoi partition. For sharply peaked sources, the algorithm will place the units directly on top of the high density ridges.

To demonstrate the generality of our non–parametric approach, we perform source separation and density coding of a non-linear mixture. Because our network has local dynamics, with enough units, the network can follow the curved "independent component contours" of the input distribution. The result is shown in Fig. (2).

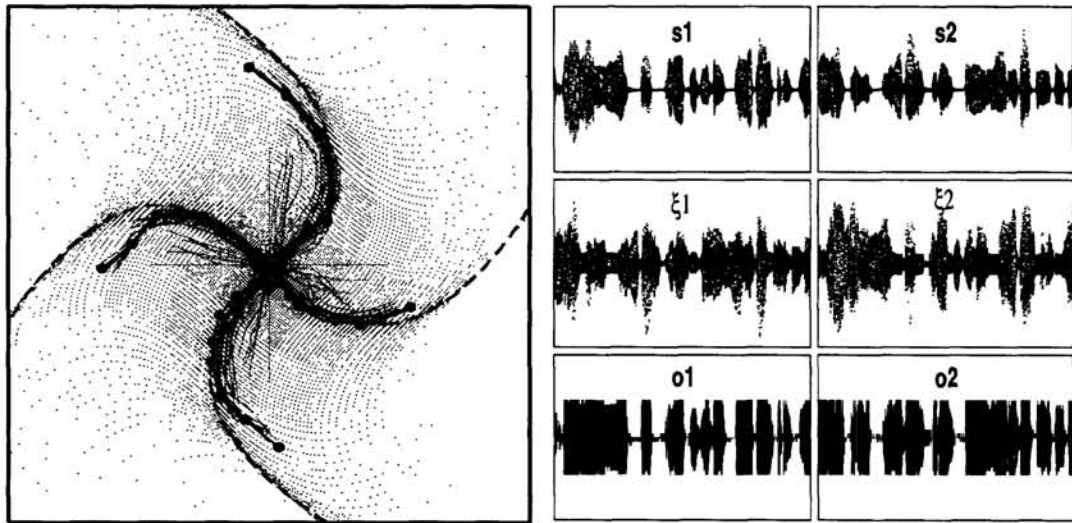

Figure 2: Source separation of non–linear mixture. The mixture is given by $\xi_1 = -2sgn(s_1) \cdot s_1^2 + 1.1s_1 - s_2$, $\xi_2 = -2sgn(s_2) \cdot s_2^2 + s_1 + 1.1s_2$. Left: the SOM configuration is shown periodically in the figure, with the configuration after 12000 points indicated by the dots. Dashed lines denote two independent component contours. Right: the sources ($s_1$, $s_2$), mixtures ($\xi_1$, $\xi_2$) and pseudo–histogram–equalized representations ($o_1$, $o_2$).

To unmix the input, a parametric separation approach can be taken where least squares fit to the branch contours is used. For the source separation in Fig. (1a), assuming linear mixing and inserting the branch coordinate system into an unmixing matrix, we find a reduction of the amplitudes of the mixtures to less than one percent of the signal. This is typical of the quality of separation obtained in our simulations. For the non–linear source separation in Fig. (2), parametric unmixing can similarly be accomplished by least squares fit to polynomial contours with

quadratic terms. Alternatively, taking full advantage of the non–parametric nature of the SOM approach, an approximation of the independent sources can be constructed from the positions $\vec{w}_{i^*}$ of the winning unit. Or as we show in Fig. (2b), the cell labels $i^*$ can be used to give a pseudo–histogram–equalized source representation. This non–parametric approach is thus much more general in the sense that no model is needed of the mixing transformation. Since there is only one winning unit along one branch, only one output channel is active at any given time. For sharply peaked source distributions such as speech, this does not significantly hinder the fidelity of the source representation since the input sources hover around zero most of the time. This property also has the potential for utilization in compression. However, for a full rigorous histogram–equalized source representation, we must turn to a network with a topology that matches the dimensionality of the input.

## 3  ARBITRARY DISTRIBUTIONS

For mixtures of sources with arbitrary distributions, we seek a full $N$ dimensional equipartition. We define an $(M, N)$ partition of $\Re^N$ to be a partition of $\Re^N$ into $(M+1)^N$ regions by $M$ parallel cuts normal to each of $N$ distinct directions. The simplest equipartition of a source mixtures is the trivial equipartition along the independent component axes (ICA). Our goal is to achieve this trivial ICA aligned equipartition using a hypercube architecture SOM with $M+1$ units per dimension. For an $(M, N)$ equipartition, since the number of degrees of freedom to define the $MN$ hyperplanes grows quadratically in $N$, while the number of constraints grows exponentially in $N$, for large enough $M$ the desired trivial equipartition will the unique $(M, N)$ equipartition. We postulate that $M = 2$ suffices for uniqueness. Complementary to this claim, it is known that a $(1, N)$ equipartition does not exist for *arbitrary* distributions for $N \geq 5$ (Ramos 1996). The uniqueness of the $(M, N)$ equipartition of source mixtures thus provides an exact verification condition for noiseless source separation.

With $\vec{\epsilon} = \vec{i^*} - \vec{i}$, the digital equipartition learning rule is given by:

$$\Delta \vec{w}_{\vec{i}} \quad = \quad \kappa \Lambda(\vec{\epsilon}) \cdot sgn(\vec{\epsilon}) \tag{2}$$

$$\Delta \vec{w}_{i^*} \quad = \quad \sum_{\vec{i}} \Delta \vec{w}_{\vec{i}}, \tag{3}$$

where

$$\Lambda(\vec{\epsilon}) = \Lambda(-\vec{\epsilon}). \tag{4}$$

Equipartion of the input distribution can easily be shown to be a steady–state of the dynamics. Let $q_{\vec{k}}$ be the probability measure of unit $\vec{k}$. For the steady–state:

$$< \Delta \vec{w}_{\vec{k}} > \quad = \quad 0$$

$$= \quad \sum_{\vec{i}} q_{\vec{i}} \cdot \Lambda(\vec{i} - \vec{k}) \cdot sgn(\vec{i} - \vec{k}) + q_{\vec{k}} \sum_{\vec{i}} \Lambda(\vec{k} - \vec{i}) \cdot sgn(\vec{k} - \vec{i})$$

$$= \quad \sum_{\vec{i}} (q_{\vec{i}} - q_{\vec{k}}) \cdot \Lambda(\vec{i} - \vec{k}) \cdot sgn(\vec{i} - \vec{k}),$$

for all units $\vec{k}$. By inspection, equipartition, where $q_{\vec{i}} = q_{\vec{k}_0}$ for all units $\vec{i}$ is a solution to the equation above. It has been shown that equipartition is the only

steady–state of the learning rule in two dimensional rectangular SOM's (Lin and Cowan 1996), though with the highly overconstrained steady–state equations, the result should be much more general.

One further modification of the SOM is required. The desired trivial ICA equipartition is not a proper Voronoi partition except when the independent component axes are orthogonal. To obtain the desired equipartition, it is necessary to change the definition of the winning unit $\vec{i}^*$. Let

$$\Omega(\vec{w}_{\vec{i}}) = \{\vec{\xi} \in \Re^N : \vec{i}^* = \vec{i}\} \tag{5}$$

be the winning region of the unit at $\vec{w}_{\vec{i}}$. Since a histogram–equalized representation independent of the mixing transformation $A$ is desired, we require that

$$\{A\Omega(\vec{w})\} = \{\Omega(A\vec{w})\}, \tag{6}$$

i.e., $\Omega$ is *equivariant* under the action of $A$ (see e.g. Golubitsky 1988).

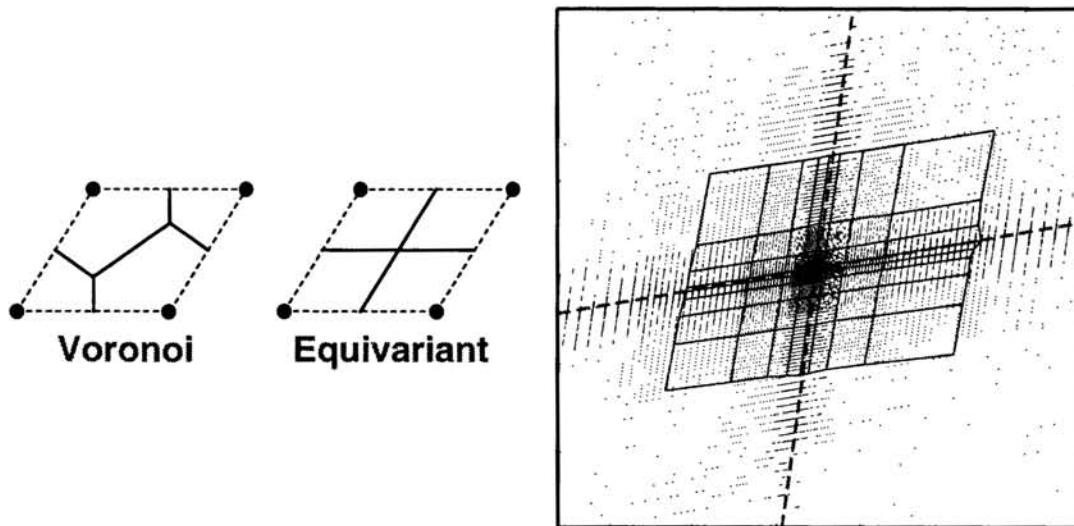

Figure 3: Left: Voronoi and equivariant partitions of the a primitive cell. Right: configuration of the SOM after 4000 points. Initially the units of the SOM were equally spaced and aligned along the two mixture coordinate directions.

In two dimensions, we modify the tessellation by dividing up a primitive cell amongst its constituent units along lines joining the midpoints of the sides. For a primitive cell composed of units at $\vec{a}$, $\vec{b}$, $\vec{c}$ and $\vec{d}$, the region of the primitive cell represented by $\vec{a}$ is the simply connected polygon defined by vertices at $\vec{a}$, $(\vec{a} + \vec{b})/2$, $(\vec{a} + \vec{d})/2$ and $(\vec{a} + \vec{b} + \vec{c} + \vec{d})/4$. The two partitions are contrasted in Fig. (3a). Our modified equivariant partition satisfies Eqn. (6) for all non–singular linear transformations.

The learning rule given above was shown to have an equipartition steady state. It remains, however, to align the partitions so that it becomes a valid $(M, N)$ partition. The addition of a local anisotropic coupling which physically, in analogy to elastic nets, might correspond to a bending modulus along the network's axes, will tend to align the partitions and enhance convergence to the desired steady state. We

supplemented the digital learning rule (Eqs. (2)-(3)) with a movement of the units towards the intersections of least squares line fits to the SOM grid.

Numerics are shown in Fig. 3b, where alignment with the independent component coordinate system and density estimation in the form of equipartition can be seen. The aligned equipartition representation formed by the network gives histogram–equalized representations of the independent sources, which, because of the equivariant nature of the SOM, will be independent of the mixing matrix.

## 4  DISCUSSION

Most source separation algorithms are parametric density estimation approaches (e.g. Bell and Sejnowski 1995, Pearlmutter and Parra 1996). Alternatively in parallel with this work, the standard SOM was used for the separation of both discrete and uniform sources (Herrmann and Yang 1996, Pajunen et. al. 1996). The source separation approach taken here is very general in the sense that no *a priori* assumptions about the individual source distributions and mixing transformation are made. Our approach's local non–parametric nature allows for source separation of non–linear mixtures and also possibly the separation of more sharply peaked sources from fewer mixtures. The low to high dimensional map required for the later task will be prohibitively difficult for parametric unmixing approaches.

For density estimation in the form of equipartition, we point out the importance of a digital scale–invariant algorithm. Direct dependence on $\vec{\xi}$ and $\vec{w}_{\vec{i}}$ has been extracted out of the learning rule. Because the update depends only upon the partition, the network learns from its own coarse response to stimuli. This along with the equivariant partition modification underscore the dynamic partition nature of the our algorithm. More direct computational geometry partitioning algorithms are currently being pursued. It is also clear that a hybrid local parametric density estimation approach will work for the separation of sharply peaked sources (Bishop et. al. 1996, Utsugi 1996).

## 5  CONCLUSIONS

We have extracted the local geometrical structure of transformations of product distributions. By modifying the SOM algorithm we developed a network with the capability of non–parametrically separating out non–linear source mixtures. Sharply peaked sources allow for quick separation via a branched SOM network. For arbitrary source distributions, we introduce the (M,N) equipartition, the uniqueness of which provides an exact verification condition for source separation.

Fundamentally, equipartition of activity is a very sensible resource allocation principle. In this work, the local equipartition coding and source separation processing proceed in tandem, resulting in optimal coding and processing of source mixtures. We believe the digital "counting" aspect of the learning rule, the learning based on the network's own coarse response to stimuli, the local nature of the dynamics, and the coupling of coding and processing make this an attractive approach from both computational and neural modeling perspectives.

## Footnotes

[1]The sign function $sgn(i)$ takes on a value of 1 for $i > 0$, 0 for $i = 0$ and $-1$ for $i < 0$. Here the sign function acts component-wise on the vector.

## References

Bauer, H.-U., Der, R., and Herrmann, M. 1996. Controlling the magnification factor of self–organizing feature maps. *Neural Comp.* **8**, 757-771.

Bell, A. J., and Sejnowski, T. J. 1995. An information-maximization approach to blind separation and blind deconvolution. *Neural Comp.* **7**,1129-1159.

Bell, A. J., and Sejnowski, T. J. 1996. Edges are the "independent components" of natural scenes. *NIPS*9*.

Bishop, C. M. and Williams, C. 1996. GTM: A principled alternative to the self–organizing map. *NIPS*9*.

Field, D. J. 1994. What is the goal of sensory coding? *Neural Comp.* **6**, 559-601.

Golubitsky, M., Stewart, I., and Schaeffer, D. G. 1988. *Singularities and Groups in Bifurcation Theory*. Springer-Verlag, Berlin.

Herrmann,M. and Yang, H. H. 1996. Perspectives and limitations of self–organizing maps in blind separation of source signals. Proc. ICONIP'96.

Hertz, J., Krogh A., and Palmer, R. G. 1991. *Introduction to the Theory of Neural Computation*. Addison-Wesley, Redwood City.

Kohonen, T. 1995. *Self-Organizing Maps*. Springer-Verlag, Berlin.

Lin, J. K. and Cowan, J. D. 1996. Faithful representation of separable input distributions. To appear in *Neural Computation*.

Olshausen, B. A. and D. J. Field 1996. Emergence of simple–cell receptive field properties by learning a sparse code for natural images. *Nature* **381**, 607-609.

Pajunen, P., Hyvarinen, A. and Karhunen, J. 1996. Nonlinear blind source separation by self–organizing maps. Proc. ICONIP'96.

Pearlmutter, B. A. and Parra, L. 1996. Maximum likelihood blind source separation: a context–sensitive generalization of ICA. *NIPS*9*.

Ramos, E. A. 1996. Equipartition of mass distributions by hyperplanes. *Discrete Comput. Geom.* **15**, 147-167.

Ritter, H., and Schulten, K. 1986. On the stationary state of Kohonen's self–organizing sensory mapping. *Biol. Cybern.*, **54**, 99-106.

Utsugi, A. 1996. Hyperparameter selection for self-organizing maps. To appear in *Neural Computation*.
